# Modelling Relational Data using Bayesian Clustered Tensor Factorization

**Ilya Sutskever**
University of Toronto
ilya@cs.utoronto.ca

**Ruslan Salakhutdinov**
MIT
rsalakhu@mit.edu

**Joshua B. Tenenbaum**
MIT
jbt@mit.edu

## Abstract

We consider the problem of learning probabilistic models for complex relational structures between various types of objects. A model can help us "understand" a dataset of relational facts in at least two ways, by finding interpretable structure in the data, and by supporting predictions, or inferences about whether particular unobserved relations are likely to be true. Often there is a tradeoff between these two aims: cluster-based models yield more easily interpretable representations, while factorization-based approaches have given better predictive performance on large data sets. We introduce the Bayesian Clustered Tensor Factorization (BCTF) model, which embeds a factorized representation of relations in a nonparametric Bayesian clustering framework. Inference is fully Bayesian but scales well to large data sets. The model simultaneously discovers interpretable clusters and yields predictive performance that matches or beats previous probabilistic models for relational data.

## 1 Introduction

Learning with relational data, or sets of propositions of the form (object, relation, object), has been important in a number of areas of AI and statistical data analysis. AI researchers have proposed that by storing enough everyday relational facts and generalizing appropriately to unobserved propositions, we might capture the essence of human common sense. For instance, given propositions such as (cup, used-for, drinking), (cup, can-contain, juice), (cup, can-contain, water), (cup, can-contain, coffee), (glass, can-contain, juice), (glass, can-contain, water), (glass, can-contain, wine), and so on, we might also infer the propositions (glass, used-for, drinking), (glass, can-contain, coffee), and (cup, can-contain, wine). Modelling relational data is also important for more immediate applications, including problems arising in social networks [2], bioinformatics [16], and collaborative filtering [18].

We approach these problems using probabilistic models that define a joint distribution over the truth values of all conceivable relations. Such a model defines a joint distribution over the binary variables $T(a, r, b) \in \{0, 1\}$, where $a$ and $b$ are objects, $r$ is a relation, and the variable $T(a, r, b)$ determines whether the relation $(a, r, b)$ is true. Given a set of true relations $S = \{(a, r, b)\}$, the model predicts that a new relation $(a, r, b)$ is true with probability $P(T(a, r, b) = 1|S)$.

In addition to making predictions on new relations, we also want to understand the data—that is, to find a small set of interpretable laws that explains a large fraction of the observations. By introducing hidden variables over simple hypotheses, the posterior distribution over the hidden variables will concentrate on the laws the data is likely to obey, while the nature of the laws depends on the model. For example, the Infinite Relational Model (IRM) [8] represents simple laws consisting of partitions of objects and partitions of relations. To decide whether the relation $(a, r, b)$ is valid, the IRM simply checks that the clusters to which $a$, $r$, and $b$ belong are compatible. The main advantage of the IRM is its ability to extract meaningful partitions of objects and relations from the observational data,

which greatly facilitates exploratory data analysis. More elaborate proposals consider models over more powerful laws (e.g., first order formulas with noise models or multiple clusterings), which are currently less practical due to the computational difficulty of their inference problems [7, 6, 9].

Models based on matrix or tensor factorization [18, 19, 3] have the potential of making better predictions than interpretable models of similar complexity, as we demonstrate in our experimental results section. Factorization models learn a distributed representation for each object and each relation, and make predictions by taking appropriate inner products. Their strength lies in the relative ease of their continuous (rather than discrete) optimization, and in their excellent predictive performance. However, it is often hard to understand and analyze the learned latent structure.

The tension between interpretability and predictive power is unfortunate: it is clearly better to have a model that has both strong predictive power and interpretability. We address this problem by introducing the Bayesian Clustered Tensor Factorization (BCTF) model, which combines good interpretability with excellent predictive power. Specifically, similarly to the IRM, the BCTF model learns a partition of the objects and a partition of the relations, so that the truth-value of a relation $(a, r, b)$ depends primarily on the compatibility of the clusters to which $a$, $r$, and $b$ belong. At the same time, every entity has a distributed representation: each object $a$ is assigned the two vectors $\mathbf{a}_L, \mathbf{a}_R$ (one for $a$ being a left argument in a relation and one for it being a right argument), and a relation $r$ is assigned the matrix $\mathbf{R}$. Given the distributed representations, the truth of a relation $(a, r, b)$ is determined by the value of $\mathbf{a}_L^\top \mathbf{R} \mathbf{b}_R$, while the object partition encourages the objects within a cluster to have similar distributed representations (and similarly for relations).

The experiments show that the BCTF model achieves better predictive performance than a number of related probabilistic relational models, including the IRM, on several datasets. The model is scalable, and we apply it on the Movielens [15] and the Conceptnet [10] datasets. We also examine the structure found in BCTF's clusters and learned vectors. Finally, our results provide an example where the performance of a Bayesian model *substantially* outperforms a corresponding MAP estimate for large sparse datasets with minimal manual hyperparameter selection.

## 2 The Bayesian Clustered Tensor Factorization (BCTF)

We begin with a simple tensor factorization model. Suppose that we have a fixed finite set of objects $O$ and a fixed finite set of relations $R$. For each object $a \in O$ the model maintains two vectors $\mathbf{a}_L, \mathbf{a}_R \in \mathbb{R}^d$ (the left and the right arguments of the relation), and for each relation $r \in R$ it maintains a matrix $\mathbf{R} \in \mathbb{R}^{d \times d}$, where $d$ is the dimensionality of the model. Given a setting of these parameters (collectively denoted by $\theta$), the model independently chooses the truth-value of each relation $(a, r, b)$ from the distribution $P(T(a, r, b) = 1|\theta) = 1/(1 + \exp(-\mathbf{a}_L^\top \mathbf{R} \mathbf{b}_R))$. In particular, given a set of known relations $S$, we can learn the parameters by maximizing a penalized log likelihood $\log P(S|\theta) - Reg(\theta)$. The necessity of having a pair of parameters $\mathbf{a}_L, \mathbf{a}_R$, instead of a single distributed representation $\mathbf{a}$, will become clear later.

Next, we define a prior over the vectors $\{\mathbf{a}_L\}$, $\{\mathbf{a}_R\}$, and $\{\mathbf{R}\}$. Specifically, the model defines a prior distribution over partitions of objects and partitions of relations using the Chinese Restaurant Process. Once the partitions are chosen, each cluster $C$ samples its own prior mean and prior diagonal covariance, which are then used to independently sample vectors $\{\mathbf{a}_L, \mathbf{a}_R : a \in C\}$ that belong to cluster $C$ (and similarly for the relations, where we treat $\mathbf{R}$ as a $d^2$-dimensional vector). As a result, objects within a cluster have similar distributed representations. When the clusters are sufficiently tight, the value of $\mathbf{a}_L^\top \mathbf{R} \mathbf{b}_R$ is mainly determined by the clusters to which $a$, $r$, and $b$ belong. At the same time, the distributed representations help generalization, because they can represent graded similarities between clusters and fine differences between objects in the same cluster. Thus, given a set of relations, we expect the model to find both meaningful clusters of objects and relations, as well as predictive distributed representations.

More formally, assume that $O = \{a_1, \ldots, a_N\}$ and $R = \{r_1, \ldots, r_M\}$. The model is defined as follows:

$$P(\text{obs}, \theta, c, \alpha, \alpha_{DP}) = P(\text{obs}|\theta, \sigma^2)P(\theta|c, \alpha)P(c|\alpha_{DP})P(\alpha_{DP}, \alpha, \sigma^2) \qquad (1)$$

where the observed data obs is a set of triples and their truth values $\{(a, r, b), t\}$; the variable $c = \{c_{obj}, c_{rel}\}$ contains the cluster assignments (partitions) of the objects and the relations; the variable $\theta = \{\mathbf{a}_L, \mathbf{a}_R, \mathbf{R}\}$ consists of the distributed representations of the objects and the relations, and

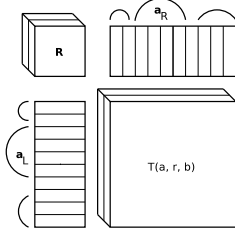

Figure 1: A schematic diagram of the model, where the arcs represent the object clusters and the vectors within each cluster are similar. The model predicts $T(a, r, b)$ with $\mathbf{a}_L^\top \mathbf{R} \mathbf{b}_R$.

$\{\sigma^2, \alpha, \alpha_{DP}\}$ are the model hyperparameters. Two of the above terms are given by

$$P(\text{obs}|\theta) \quad = \prod_{\{(a,r,b),t\}\in\text{obs}} \mathcal{N}(t|\mathbf{a}_L^\top \mathbf{R} \mathbf{b}_R, \sigma^2) \tag{2}$$

$$P(c|\alpha_{DP}) \quad = \quad CRP(c_{obj}|\alpha_{DP})CRP(c_{rel}|\alpha_{DP}) \tag{3}$$

where $\mathcal{N}(t|\mu, \sigma^2)$ denotes the Gaussian distribution with mean $\mu$ and variance $\sigma^2$, and $CRP(c|\alpha)$ denotes the probability of the partition induced by $c$ under the Chinese Restaurant Process with concentration parameter $\alpha$. The Gaussian likelihood in Eq. 2 is far from ideal for modelling binary data, but, similarly to [19, 18], we use it instead of the logistic function because it makes the model conjugate and Gibbs sampling easier.

Defining $P(\theta|c, \alpha)$ takes a little more work. Given the partitions, the sets of parameters $\{\mathbf{a}_L\}$, $\{\mathbf{a}_R\}$, and $\{\mathbf{R}\}$ become independent, so

$$P(\theta|c, \alpha) = P(\{\mathbf{a}_L\}|c_{obj}, \alpha_{obj})P(\{\mathbf{a}_R\}|c_{obj}, \alpha_{obj})P(\{\mathbf{R}\}|c_{rel}, \alpha_{rel}) \tag{4}$$

The distribution over the relation-vectors is given by

$$P(\{\mathbf{R}\}|c_{rel}, \alpha_{rel}) = \prod_{k=1}^{|c_{rel}|} \int_{\mu,\Sigma} \prod_{i:c_{rel,i}=k} \mathcal{N}(\mathbf{R}_i|\mu, \Sigma)\, dP(\mu, \Sigma|\alpha_{rel}) \tag{5}$$

where $|c_{rel}|$ is the number of clusters in the partition $c_{rel}$. This is precisely a Dirichlet process mixture model [13]. We further place a Gaussian-Inverse-Gamma prior over $(\mu, \Sigma)$:

$$P(\mu, \Sigma|\alpha_{rel}) \quad = \quad P(\mu|\Sigma)P(\Sigma|\alpha_{rel}) = \mathcal{N}(\mu|0, \Sigma)\prod_{d'} IG(\sigma^2_{d'}|\alpha_{rel}, 1) \tag{6}$$

$$\propto \quad \exp\left(-\sum_{d'} \frac{\mu^2_{d'}/2+1}{\sigma^2_{d'}}\right)\prod_{d'} (\sigma^2_{d'})^{-0.5-\alpha_{rel}-1} \tag{7}$$

where $\Sigma$ is a diagonal matrix whose entries are $\sigma^2_{d'}$, the variable $d'$ ranges over the dimensions of $\mathbf{R}_i$ (so $1 \le d' \le d^2$), and $IG(x|\alpha, \beta)$ denotes the inverse-Gamma distribution with shape parameter $\alpha$ and scale parameter $\beta$. This prior makes many useful expectations analytically computable. The terms $P(\{\mathbf{a}_L\}|c_{obj}, \alpha_{obj})$ and $P(\{\mathbf{a}_R\}|c_{obj}, \alpha_{obj})$ are defined analogously to Eq. 5.

Finally, we place an improper $P(x) \propto x^{-1}$ scale-uniform prior over each hyperparameter independently.

**Inference**

We now briefly describe the MCMC algorithm used for inference. Before starting the Markov chain, we find a MAP estimate of the model parameters using the method of conjugate gradient (but we do not optimize over the partitions). The MAP estimate is then used to initialize the Markov chain. Each step of the Markov chain consists of a number of internal steps. First, given the parameters $\theta$, the chain updates $c = (c_{rel}, c_{obj})$ using a collapsed Gibbs sampling sweep and a step of the split-and-merge algorithm (where the launch state was obtained with two sweeps of Gibbs sampling starting from a uniformly random cluster assignment) [5]. Next, it samples from the posterior mean

and covariance of each cluster, which is the distribution proportional to the term being integrated in Eq. 5.

Next, the Markov chain samples the parameters $\{\mathbf{a}_L\}$ given $\{\mathbf{a}_R\}$, $\{\mathbf{R}\}$, and the cluster posterior means and covariances. This step is tractable since the conditional distribution over the object vectors $\{\mathbf{a}_L\}$ is Gaussian and factorizes into the product of conditional distributions over the individual object vectors. This conditional independence is important, since it tends to make the Markov chain mix faster, and is a direct consequence of each object $a$ having two vectors, $\mathbf{a}_L$ and $\mathbf{a}_R$. If each object $a$ was only associated with a single vector $\mathbf{a}$ (and not $\mathbf{a}_L, \mathbf{a}_R$), the conditional distribution over $\{\mathbf{a}\}$ would not factorize, which in turn would require the use of a slower sequential Gibbs sampler. In the current setting, we can further speed up the inference by sampling from conditional distributions in parallel. The speedup could be substantial, particularly when the number of objects is large. The disadvantage of using two vectors for each object is that the model cannot as easily capture the "position-independent" properties of the object, especially in the sparse regime.

Sampling $\{\mathbf{a}_L\}$ from the Gaussian takes time proportional to $d^3 \cdot N$, where $N$ is the number of objects. While we do the same for $\{\mathbf{a}_R\}$, we run a standard hybrid Monte Carlo to update the matrices $\{\mathbf{R}\}$ using 10 leapfrog steps of size $10^{-5}$ [12]. Each matrix, which we treat as a vector, has $d^2$ dimensions, so direct sampling from the Gaussian distribution scales as $d^6 \cdot M$, which is slow even for small values of $d$ (e.g. 20). Finally, we make a small symmetric multiplicative change to each hyperparameter and accept or reject its new value according to the Metropolis-Hastings rule.

# 3 Evaluation

In this section, we show that the BCTF model has excellent predictive power and that it finds interpretable clusters by applying it to five datasets and comparing its performance to the IRM [8] and the Multiple Relational Clustering (MRC) model [9]. We also compare BCTF to its simpler counterpart: a Bayesian Tensor Factorization (BTF) model, where all the objects and the relations belong to a single cluster. The Bayesian Tensor Factorization model is a generalization of the Bayesian probabilistic matrix factorization [17], and is closely related to many other existing tensor-factorization methods [3, 14, 1]. In what follows, we will describe the datasets, report the predictive performance of our and of the competing algorithms, and examine the structure discovered by BCTF.

## 3.1 Description of the Datasets

We use three of the four datasets used by [8] and [9], namely, the Animals, the UML, and the Kinship dataset, as well the Movielens [15] and the Conceptnet datasets [10].

1. The animals dataset consists of 50 animals and 85 binary attributes. The dataset is a fully observed matrix—so there is only one relation.

2. The kinship dataset consists of kinship relationships among the members of the Alyawarra tribe [4]. The dataset contains 104 people and 26 relations. This dataset is dense and has $104 \cdot 26 \cdot 104 = 218216$ observations, most of which are 0.

3. The UML dataset [11] consists of a 135 medical terms and 49 relations. The dataset is also fully observed and has $135 \cdot 49 \cdot 135 = 893025$ (mostly 0) observations.

4. The Movielens [15] dataset consists of 1000209 observed integer ratings of 6041 movies on a scale from 1 to 5, which are rated by 3953 users. The dataset is 95.8% sparse.

5. The Conceptnet dataset [10] is a collection of common-sense assertions collected from the web. It consists of about 112135 "common-sense" assertions such as (hockey, is-a, sport). There are 19 relations and 17571 objects. To make our experiments faster, we used only the 7000 most frequent objects, which resulted in 82062 true facts. For the negative data, we sampled twice as many random object-relation-object triples and used them as the false facts. As a result, there were 246186 binary observations in this dataset. The dataset is 99.9% sparse.

## 3.2 Experimental Protocol

To facilitate comparison with [9], we conducted our experiments the following way. First, we normalized each dataset so the mean of its observations was 0. Next, we created 10 random train/test

| | animals | | kinship | | UML | | movielens | | conceptnet | |
|---|---|---|---|---|---|---|---|---|---|---|
| algorithm | RMSE | AUC | RMSE | AUC | RMSE | AUC | RMSE | AUC | RMSE | AUC |
| $MAP_{20}$ | 0.467 | 0.78 | 0.122 | 0.82 | 0.033 | 0.96 | 0.899 | – | 0.536 | 0.57 |
| $MAP_{40}$ | 0.528 | 0.68 | 0.110 | **0.90** | **0.024** | **0.98** | 0.933 | – | 0.614 | 0.48 |
| $BTF_{20}$ | 0.337 | 0.85 | 0.122 | 0.82 | 0.033 | 0.96 | 0.835 | – | 0.275 | 0.93 |
| $BCTF_{20}$ | **0.331** | **0.86** | 0.122 | 0.82 | 0.033 | 0.96 | 0.836 | – | 0.278 | 0.93 |
| $BTF_{40}$ | 0.338 | **0.86** | **0.108** | **0.90** | **0.024** | **0.98** | **0.834** | – | 0.267 | **0.94** |
| $BCTF_{40}$ | 0.336 | **0.86** | **0.108** | **0.90** | **0.024** | **0.98** | 0.836 | – | **0.260** | **0.94** |
| IRM [8] | 0.382 | 0.75 | 0.140 | 0.66 | 0.054 | 0.70 | – | – | – | – |
| MRC [9] | – | 0.81 | – | 0.85 | – | **0.98** | – | – | – | – |

Table 1: A quantitative evaluation of the algorithms using 20 and 40 dimensional vectors. We report the performance of the following algorithms: the MAP-based Tensor Factorization, the Bayesian Tensor Factorization (BTF) with MCMC (where all objects belong to a single cluster), the full Bayesian Clustered Tensor Factorization (BCTF), the IRM [8] and the MRC [9].

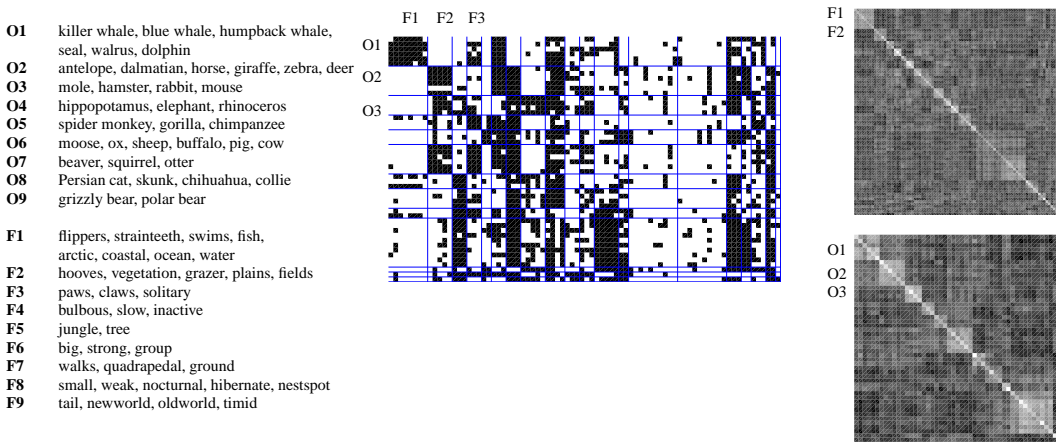

| | |
|---|---|
| **O1** | killer whale, blue whale, humpback whale, seal, walrus, dolphin |
| **O2** | antelope, dalmatian, horse, giraffe, zebra, deer |
| **O3** | mole, hamster, rabbit, mouse |
| **O4** | hippopotamus, elephant, rhinoceros |
| **O5** | spider monkey, gorilla, chimpanzee |
| **O6** | moose, ox, sheep, buffalo, pig, cow |
| **O7** | beaver, squirrel, otter |
| **O8** | Persian cat, skunk, chihuahua, collie |
| **O9** | grizzly bear, polar bear |
| | |
| **F1** | flippers, strainteeth, swims, fish, arctic, coastal, ocean, water |
| **F2** | hooves, vegetation, grazer, plains, fields |
| **F3** | paws, claws, solitary |
| **F4** | bulbous, slow, inactive |
| **F5** | jungle, tree |
| **F6** | big, strong, group |
| **F7** | walks, quadrapedal, ground |
| **F8** | small, weak, nocturnal, hibernate, nestspot |
| **F9** | tail, newworld, oldworld, timid |

Figure 2: Results on the Animals dataset. **Left:** The discovered clusters. **Middle:** The biclustering of the features. **Right:** The covariance of the distributed representations of the animals (bottom) and their attributes (top).

splits, where 10% of the data was used for testing. For the Conceptnet and the Movielens datasets, we used only two train/test splits and at most 30 clusters, which made our experiments faster. We report test root mean squared error (RMSE) and the area under the precision recall curve (AUC) [9]. For the IRM[1] we make predictions as follows. The IRM partitions the data into blocks; we compute the smoothed mean of the observed entries of each block and use it to predict the test entries in the same block.

### 3.3 Results

We first applied BCTF to the Animals, Kinship, and the UML datasets using 20 and 40-dimensional vectors. Table 1 shows that BCTF substantially outperforms IRM and MRC in terms of both RMSE and AUC. In fact, for the Kinship and the UML datasets, the simple tensor factorization model trained by MAP performs as well as BTF and BCTF. This happens because for these datasets the number of observations is much larger than the number of parameters, so there is little uncertainty about the true parameter values. However, the Animals dataset is considerably smaller, so BTF performs better, and BCTF performs even better than the BTF model.

We then applied BCTF to the Movielens and the Conceptnet datasets. We found that the MAP estimates suffered from significant overfitting, and that the fully Bayesian models performed much better. This is important because both datasets are sparse, which makes overfitting difficult to combat. For the extremely sparse Conceptnet dataset, the BCTF model further improved upon simpler

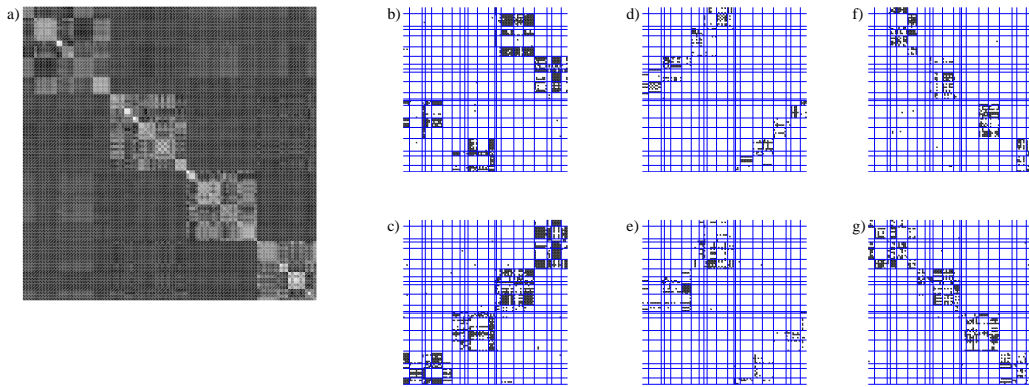

Figure 3: Results on the Kinship dataset. **Left:** The covariance of the distributed representations $\{\mathbf{a}_L\}$ learned for each person. **Right:** The biclustering of a subset of the relations.

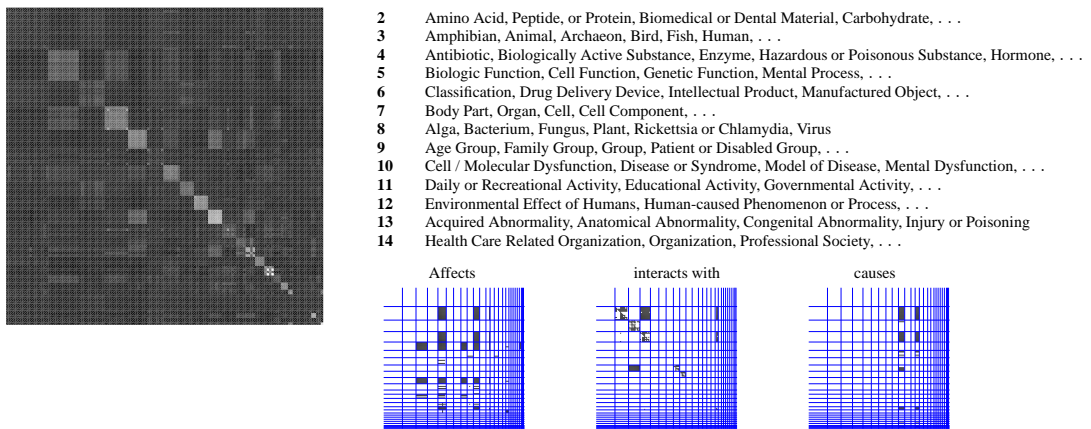

| 2 | Amino Acid, Peptide, or Protein, Biomedical or Dental Material, Carbohydrate, . . . |
| 3 | Amphibian, Animal, Archaeon, Bird, Fish, Human, . . . |
| 4 | Antibiotic, Biologically Active Substance, Enzyme, Hazardous or Poisonous Substance, Hormone, . . . |
| 5 | Biologic Function, Cell Function, Genetic Function, Mental Process, . . . |
| 6 | Classification, Drug Delivery Device, Intellectual Product, Manufactured Object, . . . |
| 7 | Body Part, Organ, Cell, Cell Component, . . . |
| 8 | Alga, Bacterium, Fungus, Plant, Rickettsia or Chlamydia, Virus |
| 9 | Age Group, Family Group, Group, Patient or Disabled Group, . . . |
| 10 | Cell / Molecular Dysfunction, Disease or Syndrome, Model of Disease, Mental Dysfunction, . . . |
| 11 | Daily or Recreational Activity, Educational Activity, Governmental Activity, . . . |
| 12 | Environmental Effect of Humans, Human-caused Phenomenon or Process, . . . |
| 13 | Acquired Abnormality, Anatomical Abnormality, Congenital Abnormality, Injury or Poisoning |
| 14 | Health Care Related Organization, Organization, Professional Society, . . . |

Figure 4: Results on the medical UML dataset. **Left:** The covariance of the distributed representations $\{\mathbf{a}_L\}$ learned for each object. **Right:** The inferred clusters, along with the biclustering of a subset of the relations.

BTF model. We do not report results for the IRM, because the existing off-the-shelf implementation could not handle these large datasets.

We now examine the latent structure discovered by the BCTF model by inspecting a sample produced by the Markov chain. Figure 2 shows some of the clusters learned by the model on the Animals dataset. It also shows the biclustering, as well as the covariance of the distributed representations of the animals and their attributes, sorted by their clusters. By inspecting the covariance, we can determine the clusters that are tight and the affinities between the clusters. Indeed, the cluster structure is reflected in the block-diagonal structure of the covariance matrix. For example, the covariance of the attributes (see Fig. 2, top-right panel) shows that cluster F1, containing {flippers, stainteeth,swims} is similar to cluster F4, containing {bulbous, slow, inactive}, but is very dissimilar to F2, containing {hooves, vegetation, grazer}.

Figure 3 displays the learned representation for the Kinship dataset. The kinship dataset has 104 people with complex relationships between them: each person belongs to one of four sections, which strongly constrains the other relations. For example, a person in section 1 has a father in section 3 and a mother in section 4 (see [8, 4] for more details). After learning, each cluster was almost completely localized in gender, section, and age. For clarity of presentation, we sort the clusters first by their section, then by their gender, and finally by their age, as done in [8]. Figure 3 (panels (b-g)) displays some of the relations according to this clustering, and panel (a) shows the covariance between the vectors $\{\mathbf{a}_L\}$ learned for each person. The four sections are clearly visible in the covariance structure of the distributed representations.

Figure 4 shows the inferred clusters for the medical UML dataset. For example, the model discovers that {Amino Acid, Peptide, Protein} Affects {Biologic Function, Cell Function, Genetic Function},

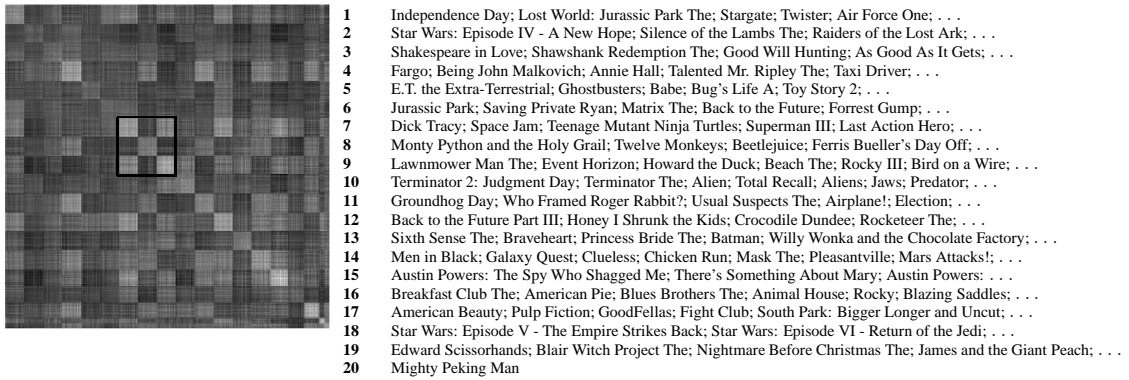

| | |
|---|---|
| 1 | Independence Day; Lost World: Jurassic Park The; Stargate; Twister; Air Force One; . . . |
| 2 | Star Wars: Episode IV - A New Hope; Silence of the Lambs The; Raiders of the Lost Ark; . . . |
| 3 | Shakespeare in Love; Shawshank Redemption The; Good Will Hunting; As Good As It Gets; . . . |
| 4 | Fargo; Being John Malkovich; Annie Hall; Talented Mr. Ripley The; Taxi Driver; . . . |
| 5 | E.T. the Extra-Terrestrial; Ghostbusters; Babe; Bug's Life A; Toy Story 2; . . . |
| 6 | Jurassic Park; Saving Private Ryan; Matrix The; Back to the Future; Forrest Gump; . . . |
| 7 | Dick Tracy; Space Jam; Teenage Mutant Ninja Turtles; Superman III; Last Action Hero; . . . |
| 8 | Monty Python and the Holy Grail; Twelve Monkeys; Beetlejuice; Ferris Bueller's Day Off; . . . |
| 9 | Lawnmower Man The; Event Horizon; Howard the Duck; Beach The; Rocky III; Bird on a Wire; . . . |
| 10 | Terminator 2: Judgment Day; Terminator The; Alien; Total Recall; Aliens; Jaws; Predator; . . . |
| 11 | Groundhog Day; Who Framed Roger Rabbit?; Usual Suspects The; Airplane!; Election; . . . |
| 12 | Back to the Future Part III; Honey I Shrunk the Kids; Crocodile Dundee; Rocketeer The; . . . |
| 13 | Sixth Sense The; Braveheart; Princess Bride The; Batman; Willy Wonka and the Chocolate Factory; . . . |
| 14 | Men in Black; Galaxy Quest; Clueless; Chicken Run; Mask The; Pleasantville; Mars Attacks!; . . . |
| 15 | Austin Powers: The Spy Who Shagged Me; There's Something About Mary; Austin Powers: . . . |
| 16 | Breakfast Club The; American Pie; Blues Brothers The; Animal House; Rocky; Blazing Saddles; . . . |
| 17 | American Beauty; Pulp Fiction; GoodFellas; Fight Club; South Park: Bigger Longer and Uncut; . . . |
| 18 | Star Wars: Episode V - The Empire Strikes Back; Star Wars: Episode VI - Return of the Jedi; . . . |
| 19 | Edward Scissorhands; Blair Witch Project The; Nightmare Before Christmas The; James and the Giant Peach; . . . |
| 20 | Mighty Peking Man |

Figure 5: Results on the Movielens dataset. **Left:** The covariance between the movie vectors. **Right:** The inferred clusters.

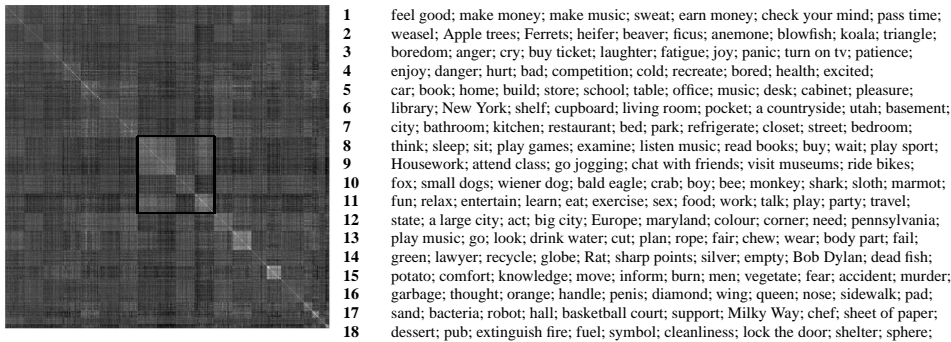

| | |
|---|---|
| 1 | feel good; make money; make music; sweat; earn money; check your mind; pass time; |
| 2 | weasel; Apple trees; Ferrets; heifer; beaver; ficus; anemone; blowfish; koala; triangle; |
| 3 | boredom; anger; cry; buy ticket; laughter; fatigue; joy; panic; turn on tv; patience; |
| 4 | enjoy; danger; hurt; bad; competition; cold; recreate; bored; health; excited; |
| 5 | car; book; home; build; store; school; table; office; music; desk; cabinet; pleasure; |
| 6 | library; New York; shelf; cupboard; living room; pocket; a countryside; utah; basement; |
| 7 | city; bathroom; kitchen; restaurant; bed; park; refrigerate; closet; street; bedroom; |
| 8 | think; sleep; sit; play games; examine; listen music; read books; buy; wait; play sport; |
| 9 | Housework; attend class; go jogging; chat with friends; visit museums; ride bikes; |
| 10 | fox; small dogs; wiener dog; bald eagle; crab; boy; bee; monkey; shark; sloth; marmot; |
| 11 | fun; relax; entertain; learn; eat; exercise; sex; food; work; talk; play; party; travel; |
| 12 | state; a large city; act; big city; Europe; maryland; colour; corner; need; pennsylvania; |
| 13 | play music; go; look; drink water; cut; plan; rope; fair; chew; wear; body part; fail; |
| 14 | green; lawyer; recycle; globe; Rat; sharp points; silver; empty; Bob Dylan; dead fish; |
| 15 | potato; comfort; knowledge; move; inform; burn; men; vegetate; fear; accident; murder; |
| 16 | garbage; thought; orange; handle; penis; diamond; wing; queen; nose; sidewalk; pad; |
| 17 | sand; bacteria; robot; hall; basketball court; support; Milky Way; chef; sheet of paper; |
| 18 | dessert; pub; extinguish fire; fuel; symbol; cleanliness; lock the door; shelter; sphere; |

Figure 6: Results on the Conceptnet dataset. **Left:** The covariance of the learned $\{a_L\}$ vectors for each object. **Right:** The inferred clusters.

which is also similar, according to the covariance, to {Cell Dysfunction, Disease, Mental Dysfunction}. Qualitatively, the clustering appears to be on par with that of the IRM on all the datasets, but the BCTF model is able to predict held-out relations much better.

Figures 5 and 6 display the learned clusters for the Movielens and the Conceptnet datasets. For the Movielens dataset, we show the most frequently-rated movies in each cluster where the clusters are sorted by size. We also show the covariance between the movie vectors which are sorted by the clusters, where we display only the 100 most frequently-rated movies per cluster. The covariance matrix is aligned with the table on the right, making it easy to see how the clusters relate to each other. For example, according to the covariance structure, clusters 7 and 9, containing Hollywood action/adventure movies are similar to each other but are dissimilar to cluster 8, which consists of comedy/horror movies.

For the Conceptnet dataset, Fig. 6 displays the 100 most frequent objects per category. From the covariance matrix, we can infer that clusters 8, 9, and 11, containing concepts associated with humans taking actions, are very similar to each other, and are very dissimilar to cluster 10, which contains animals. Observe that some clusters (e.g., clusters 2-6) are not crisp, which is reflected in the smaller covariances between vectors in each of these clusters.

## 4 Discussions and Conclusions

We introduced a new method for modelling relational data which is able to both discover meaningful structure and generalize well. In particular, our results illustrate the predictive power of distributed representations when applied to modelling relational data, since even simple tensor factorization models can sometimes outperform the more complex models. Indeed, for the kinship and the UML datasets, the performance of the MAP-based tensor factorization was as good as the performance of the BCTF model, which is due to the density of these datasets: the number of observations was much larger than the number of parameters. On the other hand, for large sparse datasets, the BCTF

model significantly outperformed its MAP counterpart, and in particular, it noticeably outperformed BTF on the Conceptnet dataset.

A surprising aspect of the Bayesian model is the ease with which it worked after automatic hyperparameter selection was implemented. Furthermore, the model performs well even when the initial MAP estimate is very poor, as was the case for the 40-dimensional models on the Conceptnet dataset. This is particularly important for large sparse datasets, since finding a good MAP estimate requires careful cross-validation to select the regularization hyperparameters. Careful hyperparameter selection can be very labour-expensive because it requires careful training of a large number of models.

**Acknowledgments**

The authors acknowledge the financial support from NSERC, Shell, NTT Communication Sciences Laboratory, AFOSR FA9550-07-1-0075, and AFOSR MURI.

## Footnotes

[1]The code is available at http://www.psy.cmu.edu/~ckemp/code/irm.html

# References

[1] Edoardo Airoldi, David M. Blei, Stephen E. Fienberg, and Eric P. Xing. Mixed membership stochastic blockmodels. In *NIPS*, pages 33–40. MIT Press, 2008.

[2] P.J. Carrington, J. Scott, and S. Wasserman. *Models and methods in social network analysis*. Cambridge University Press, 2005.

[3] W. Chu and Z. Ghahramani. Probabilistic models for incomplete multi-dimensional arrays. In *Proceedings of the International Conference on Artificial Intelligence and Statistics*, volume 5, 2009.

[4] W. Denham. *The Detection of Patterns in Alyawarra Nonverbal Behavior*. PhD thesis, Department of Anthropology, University of Washington, 1973.

[5] S. Jain and R.M. Neal. A split-merge Markov chain Monte Carlo procedure for the Dirichlet process mixture model. *Journal of Computational and Graphical Statistics*, 13(1):158–182, 2004.

[6] Y. Katz, N.D. Goodman, K. Kersting, C. Kemp, and J.B. Tenenbaum. Modeling Semantic Cognition as Logical Dimensionality Reduction. In *Proceedings of Thirtieth Annual Meeting of the Cognitive Science Society*, 2008.

[7] C. Kemp, N.D. Goodman, and J.B. Tenenbaum. Theory acquisition and the language of thought. In *Proceedings of Thirtieth Annual Meeting of the Cognitive Science Society*, 2008.

[8] C. Kemp, J.B. Tenenbaum, T.L. Griffiths, T. Yamada, and N. Ueda. Learning systems of concepts with an infinite relational model. In *Proceedings of the National Conference on Artificial Intelligence*, volume 21, page 381. Menlo Park, CA; Cambridge, MA; London; AAAI Press; MIT Press; 1999, 2006.

[9] S. Kok and P. Domingos. Statistical predicate invention. In *Proceedings of the 24th international conference on Machine learning*, pages 433–440. ACM New York, NY, USA, 2007.

[10] H. Liu and P. Singh. ConceptNeta practical commonsense reasoning tool-kit. *BT Technology Journal*, 22(4):211–226, 2004.

[11] A.T. McCray. An upper-level ontology for the biomedical domain. *Comparative and Functional Genomics*, 4(1):80–84, 2003.

[12] R.M. Neal. Probabilistic inference using Markov chain Monte Carlo methods, 1993.

[13] R.M. Neal. Markov chain sampling methods for Dirichlet process mixture models. *Journal of computational and graphical statistics*, pages 249–265, 2000.

[14] Ian Porteous, Evgeniy Bart, and Max Welling. Multi-HDP: A non parametric bayesian model for tensor factorization. In Dieter Fox and Carla P. Gomes, editors, *AAAI*, pages 1487–1490. AAAI Press, 2008.

[15] J. Riedl, J. Konstan, S. Lam, and J. Herlocker. Movielens collaborative filtering data set, 2006.

[16] J.F. Rual, K. Venkatesan, T. Hao, T. Hirozane-Kishikawa, A. Dricot, N. Li, G.F. Berriz, F.D. Gibbons, M. Dreze, N. Ayivi-Guedehoussou, et al. Towards a proteome-scale map of the human protein–protein interaction network. *Nature*, 437(7062):1173–1178, 2005.

[17] R. Salakhutdinov and A. Mnih. Bayesian probabilistic matrix factorization using Markov chain Monte Carlo. In *Proceedings of the 25th international conference on Machine learning*, pages 880–887. ACM New York, NY, USA, 2008.

[18] R. Salakhutdinov and A. Mnih. Probabilistic matrix factorization. *Advances in neural information processing systems*, 20, 2008.

[19] R. Speer, C. Havasi, and H. Lieberman. AnalogySpace: Reducing the dimensionality of common sense knowledge. In *Proceedings of AAAI*, 2008.

